# Learning the $k$ in $k$-means

**Greg Hamerly, Charles Elkan**
{ghamerly,elkan}@cs.ucsd.edu
Department of Computer Science and Engineering
University of California, San Diego
La Jolla, California    92093-0114

## Abstract

When clustering a dataset, the right number $k$ of clusters to use is often
not obvious, and choosing $k$ automatically is a hard algorithmic prob-
lem. In this paper we present an improved algorithm for learning $k$ while
clustering. The G-means algorithm is based on a statistical test for the
hypothesis that a subset of data follows a Gaussian distribution. G-means
runs $k$-means with increasing $k$ in a hierarchical fashion until the test ac-
cepts the hypothesis that the data assigned to each $k$-means center are
Gaussian. Two key advantages are that the hypothesis test does not limit
the covariance of the data and does not compute a full covariance matrix.
Additionally, G-means only requires one intuitive parameter, the stand-
ard statistical significance level $\alpha$. We present results from experiments
showing that the algorithm works well, and better than a recent method
based on the BIC penalty for model complexity. In these experiments,
we show that the BIC is ineffective as a scoring function, since it does
not penalize strongly enough the model's complexity.

## 1   Introduction and related work

Clustering algorithms are useful tools for data mining, compression, probability density es-
timation, and many other important tasks. However, most clustering algorithms require the
user to specify the number of clusters (called $k$), and it is not always clear what is the best
value for $k$. Figure 1 shows examples where $k$ has been improperly chosen. Choosing $k$ is
often an *ad hoc* decision based on prior knowledge, assumptions, and practical experience.
Choosing $k$ is made more difficult when the data has many dimensions, even when clusters
are well-separated.

Center-based clustering algorithms (in particular $k$-means and Gaussian expectation-
maximization) usually assume that each cluster adheres to a unimodal distribution, such
as Gaussian. With these methods, only one center should be used to model each subset
of data that follows a unimodal distribution. If multiple centers are used to describe data
drawn from one mode, the centers are a needlessly complex description of the data, and in
fact the multiple centers capture the truth about the subset less well than one center.

In this paper we present a simple algorithm called G-means that discovers an appropriate
$k$ using a statistical test for deciding whether to split a $k$-means center into two centers.
We describe examples and present experimental results that show that the new algorithm

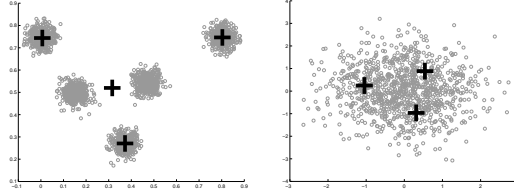

Figure 1: Two clusterings where $k$ was improperly chosen. Dark crosses are $k$-means centers. On the left, there are too few centers; five should be used. On the right, too many centers are used; one center is sufficient for representing the data. In general, one center should be used to represent one Gaussian cluster.

is successful. This technique is useful and applicable for many clustering algorithms other than $k$-means, but here we consider only the $k$-means algorithm for simplicity.

Several algorithms have been proposed previously to determine $k$ automatically. Like our method, most previous methods are wrappers around $k$-means or some other clustering algorithm for fixed $k$. Wrapper methods use splitting and/or merging rules for centers to increase or decrease $k$ as the algorithm proceeds.

Pelleg and Moore [14] proposed a regularization framework for learning $k$, which they call $X$-means. The algorithm searches over many values of $k$ and scores each clustering model using the so-called Bayesian Information Criterion [10]: $BIC(C|X) = \mathcal{L}(X|C) - \frac{p}{2}\log n$ where $\mathcal{L}(X|C)$ is the log-likelihood of the dataset $X$ according to model $C$, $p = k(d+1)$ is the number of parameters in the model $C$ with dimensionality $d$ and $k$ cluster centers, and $n$ is the number of points in the dataset. $X$-means chooses the model with the best BIC score on the data. Aside from the BIC, other scoring functions are also available.

Bischof *et al.* [1] use a minimum description length (MDL) framework, where the description length is a measure of how well the data are fit by the model. Their algorithm starts with a large value for $k$ and removes centers (reduces $k$) whenever that choice reduces the description length. Between steps of reducing $k$, they use the $k$-means algorithm to optimize the model fit to the data.

With hierarchical clustering algorithms, other methods may be employed to determine the best number of clusters. One is to build a merging tree ("dendrogram") of the data based on a cluster distance metric, and search for areas of the tree that are stable with respect to inter- and intra-cluster distances [9, Section 5.1]. This method of estimating $k$ is best applied with domain-specific knowledge and human intuition.

## 2   The Gaussian-means (G-means) algorithm

The G-means algorithm starts with a small number of $k$-means centers, and grows the number of centers. Each iteration of the algorithm splits into two those centers whose data appear not to come from a Gaussian distribution. Between each round of splitting, we run $k$-means on the entire dataset and all the centers to refine the current solution. We can initialize with just $k = 1$, or we can choose some larger value of $k$ if we have some prior knowledge about the range of $k$.

G-means repeatedly makes decisions based on a statistical test for the data assigned to each center. If the data currently assigned to a $k$-means center appear to be Gaussian, then we want to represent that data with only one center. However, if the same data do not appear

**Algorithm 1** G-means($X$, $\alpha$)
________________________________________________________
 1: Let $C$ be the initial set of centers (usually $C \leftarrow \{\bar{x}\}$).
 2: $C \leftarrow kmeans(C, X)$.
 3: Let $\{x_i | \text{class}(x_i) = j\}$ be the set of datapoints assigned to center $c_j$.
 4: Use a statistical test to detect if each $\{x_i | \text{class}(x_i) = j\}$ follow a Gaussian distribution
    (at confidence level $\alpha$).
 5: If the data look Gaussian, keep $c_j$. Otherwise replace $c_j$ with two centers.
 6: Repeat from step 2 until no more centers are added.
________________________________________________________

to be Gaussian, then we want to use multiple centers to model the data properly. The algorithm will run $k$-means multiple times (up to $k$ times when finding $k$ centers), so the time complexity is at most $O(k)$ times that of $k$-means.

The $k$-means algorithm implicitly assumes that the datapoints in each cluster are spherically distributed around the center. Less restrictively, the Gaussian expectation-maximization algorithm assumes that the datapoints in each cluster have a multidimensional Gaussian distribution with a covariance matrix that may or may not be fixed, or shared. The Gaussian distribution test that we present below are valid for either covariance matrix assumption. The test also accounts for the number of datapoints $n$ tested by incorporating $n$ in the calculation of the critical value of the test (see Equation 2). This prevents the G-means algorithm from making bad decisions about clusters with few datapoints.

## 2.1 Testing clusters for Gaussian fit

To specify the G-means algorithm fully we need a test to detect whether the data assigned to a center are sampled from a Gaussian. The alternative hypotheses are

- $H_0$: The data around the center are sampled from a Gaussian.

- $H_1$: The data around the center are not sampled from a Gaussian.

If we accept the null hypothesis $H_0$, then we believe that the one center is sufficient to model its data, and we should not split the cluster into two sub-clusters. If we reject $H_0$ and accept $H_1$, then we want to split the cluster.

The test we use is based on the Anderson-Darling statistic. This one-dimensional test has been shown empirically to be the most powerful normality test that is based on the empirical cumulative distribution function (ECDF). Given a list of values $x_i$ that have been converted to mean 0 and variance 1, let $x_{(i)}$ be the $i$th ordered value. Let $z_i = F(x_{(i)})$, where $F$ is the $N(0,1)$ cumulative distribution function. Then the statistic is

$$A^2(Z) \;=\; -\frac{1}{n}\sum_{i=1}^{n}(2i-1)\left[\log(z_i) + \log(1 - z_{n+1-i})\right] - n \tag{1}$$

Stephens [17] showed that for the case where $\mu$ and $\sigma$ are estimated from the data (as in clustering), we must correct the statistic according to

$$A_*^2(Z) \;=\; A^2(Z)(1 + 4/n - 25/(n^2)) \tag{2}$$

Given a subset of data $X$ in $d$ dimensions that belongs to center $c$, the hypothesis test proceeds as follows:

  1. Choose a significance level $\alpha$ for the test.

2. Initialize two centers, called "children" of $c$. See the text for good ways to do this.

3. Run $k$-means on these two centers in $X$. This can be run to completion, or to some early stopping point if desired. Let $c_1, c_2$ be the child centers chosen by $k$-means.

4. Let $v = c_1 - c_2$ be a $d$-dimensional vector that connects the two centers. This is the direction that $k$-means believes to be important for clustering. Then project $X$ onto $v$: $x_i' = \langle x_i, v \rangle / ||v||^2$. $X'$ is a 1-dimensional representation of the data projected onto $v$. Transform $X'$ so that it has mean 0 and variance 1.

5. Let $z_i = F(x_{(i)}')$. If $A_*^2(Z)$ is in the range of non-critical values at confidence level $\alpha$, then accept $H_0$, keep the original center, and discard $\{c_1, c_2\}$. Otherwise, reject $H_0$ and keep $\{c_1, c_2\}$ in place of the original center.

A primary contribution of this work is simplifying the test for Gaussian fit by projecting the data to one dimension where the test is simple to apply. The authors of [5] also use this approach for online dimensionality reduction during clustering. The one-dimensional representation of the data allows us to consider only the data along the direction that $k$-means has found to be important for separating the data. This is related to the problem of projection pursuit [7], where here $k$-means searches for a direction in which the data appears non-Gaussian.

We must choose the significance level of the test, $\alpha$, which is the desired probability of making a Type I error (i.e. incorrectly rejecting $H_0$). It is appropriate to use a Bonferroni adjustment to reduce the chance of making Type I errors over multiple tests. For example, if we want a 0.01 chance of making a Type I error in 100 tests, we should apply a Bonferroni adjustment to make each test use $\alpha = 0.01/100 = 0.0001$. To find $k$ final centers the G-means algorithm makes $k$ statistical tests, so the Bonferroni correction does not need to be extreme. In our tests, we always use $\alpha = 0.0001$.

We consider two ways to initialize the two child centers. Both approaches initialize with $c \pm m$, where $c$ is a center and $m$ is chosen. The first method chooses $m$ as a random $d$-dimensional vector such that $||m||$ is small compared to the distortion of the data. A second method finds the main principal component $s$ of the data (having eigenvalue $\lambda$), and chooses $m = s\sqrt{2\lambda/\pi}$. This deterministic method places the two centers in their expected locations under $H_0$. The principal component calculations require $O(nd^2 + d^3)$ time and $O(d^2)$ space, but since we only want the main principal component, we can use fast methods like the power method, which takes time that is at most linear in the ratio of the two largest eigenvalues [4]. In this paper we use principal-component-based splitting.

## 2.2 An example

Figure 2 shows a run of the G-means algorithm on a synthetic dataset with two true clusters and 1000 points, using $\alpha = 0.0001$. The critical value for the Anderson-Darling test is 1.8692 for this confidence level. Starting with one center, after one iteration of G-means, we have 2 centers and the $A_*^2$ statistic is 38.103. This is much larger than the critical value, so we reject $H_0$ and accept this split. On the next iteration, we split each new center and repeat the statistical test. The $A_*^2$ values for the two splits are 0.386 and 0.496, both of which are well below the critical value. Therefore we accept $H_0$ for both tests, and discard these splits. Thus G-means gives a final answer of $k = 2$.

## 2.3 Statistical power

Figure 3 shows the power of the Anderson-Darling test, as compared to the BIC. Lower is better for both plots. We run 1000 tests for each data point plotted for both plots. In the left

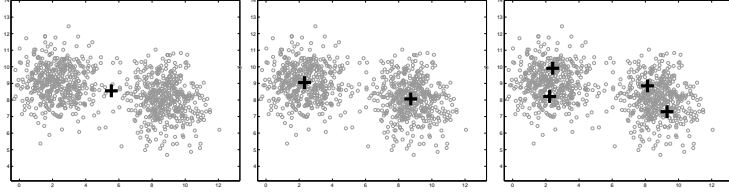

Figure 2: An example of running G-means for three iterations on a 2-dimensional dataset with two true clusters and 1000 points. Starting with one center (left plot), G-means splits into two centers (middle). The test for normality is significant, so G-means rejects $H_0$ and keeps the split. After splitting each center again (right), the test values are *not* significant, so G-means accepts $H_0$ for both tests and does not accept these splits. The middle plot is the G-means answer. See the text for further details.

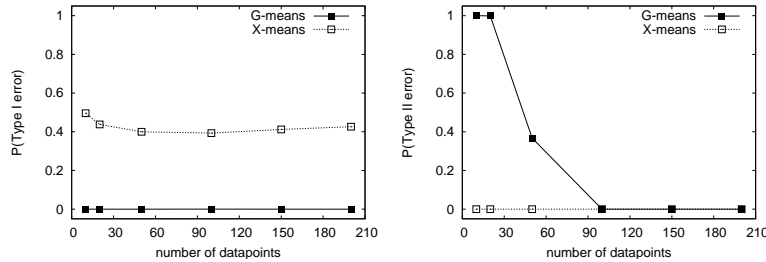

Figure 3: A comparison of the power of the Anderson-Darling test versus the BIC. For the AD test we fix the significance level ($\alpha = 0.0001$), while the BIC's significance level depends on $n$. The left plot shows the probability of incorrectly splitting (Type I error) one true 2-$d$ cluster that is 5% elliptical. The right plot shows the probability of incorrectly *not* splitting two true clusters separated by $5\sigma$ (Type II error). Both plots are functions of $n$. Both plots show that the BIC overfits (splits clusters) when $n$ is small.

plot, for each test we generate $n$ datapoints from a single true Gaussian distribution, and then plot the frequency with which BIC and G-means will choose $k = 2$ rather than $k = 1$ (i.e. commit a Type I error). BIC tends to overfit by choosing too many centers when the data is not strictly spherical, while G-means does not. This is consistent with the tests of real-world data in the next section. While G-means commits more Type II errors when $n$ is small, this prevents it from overfitting the data.

The BIC can be considered a likelihood ratio test, but with a significance level that cannot be fixed. The significance level instead varies depending on $n$ and $\Delta k$ (the change in the number of model parameters between two models). As $n$ or $\Delta k$ decrease, the significance level increases (the BIC becomes weaker as a statistical test) [10]. Figure 3 shows this effect for varying $n$. In [11] the authors show that penalty-based methods require problem-specific tuning and don't generalize as well as other methods, such as cross validation.

## 3  Experiments

Table 1 shows the results from running G-means and $X$-means on many large synthetic. On synthetic datasets with spherically distributed clusters, G-means and $X$-means do equally

Table 1: Results for many synthetic datasets. We report distortion relative to the optimum distortion for the correct clustering (closer to one is better), and time is reported relative to $k$-means run with the correct $k$. For BIC, larger values are better, but it is clear that finding the correct clustering does not always coincide with finding a larger BIC. Items with a star are where $X$-means always chose the largest number of centers we allowed.

| dataset | $d$ | method | $k$ found | distortion($\times$ optimal) | BIC($\times 10^4$) | time($\times$ $k$-means) |
|---|---|---|---|---|---|---|
| synthetic $k$=5 | 2 | G-means | **9.1$\pm$ 9.9** | **0.89$\pm$ 0.23** | -0.19$\pm$2.70 | 13.2 |
| | | $X$-means | 18.1$\pm$ 3.2 | 0.37$\pm$ 0.12 | **0.70$\pm$0.93** | **2.8** |
| synthetic $k$=20 | 2 | G-means | **20.1$\pm$ 0.6** | **0.99$\pm$ 0.01** | 0.21$\pm$0.18 | 2.1 |
| | | $X$-means | 70.5$\pm$11.6 | 9.45$\pm$28.02 | **14.83$\pm$3.50** | **1.2** |
| synthetic $k$=80 | 2 | G-means | **80.0$\pm$ 0.2** | **1.00$\pm$ 0.01** | 1.84$\pm$0.12 | 2.2 |
| | | $X$-means | 171.7$\pm$23.7 | 48.49$\pm$70.04 | **40.16$\pm$6.59** | **1.8** |
| synthetic $k$=5 | 8 | G-means | **5.0$\pm$ 0.0** | **1.00$\pm$ 0.00** | -0.74$\pm$0.16 | **4.6** |
| | | $X$-means | *20.0$\pm$ 0.0 | 0.47$\pm$ 0.03 | **-2.28$\pm$0.20** | 11.0 |
| synthetic $k$=20 | 8 | G-means | **20.0$\pm$ 0.1** | **0.99$\pm$ 0.00** | -0.18$\pm$0.17 | **2.6** |
| | | $X$-means | *80.0$\pm$ 0.0 | 0.47$\pm$ 0.01 | **14.36$\pm$0.21** | 4.0 |
| synthetic $k$=80 | 8 | G-means | **80.2$\pm$ 0.5** | **0.99$\pm$ 0.00** | 1.45$\pm$0.20 | **2.9** |
| | | $X$-means | 229.2$\pm$36.8 | 0.57$\pm$ 0.06 | **52.28$\pm$9.26** | 6.5 |
| synthetic $k$=5 | 32 | G-means | **5.0$\pm$ 0.0** | **1.00$\pm$ 0.00** | -3.36$\pm$0.21 | **4.4** |
| | | $X$-means | *20.0$\pm$ 0.0 | 0.76$\pm$ 0.00 | **-27.92$\pm$0.22** | 29.9 |
| synthetic $k$=20 | 32 | G-means | **20.0$\pm$ 0.0** | **1.00$\pm$ 0.00** | -2.73$\pm$0.22 | **2.3** |
| | | $X$-means | *80.0$\pm$ 0.0 | 0.76$\pm$ 0.01 | **-11.13$\pm$0.23** | 21.2 |
| synthetic $k$=80 | 32 | G-means | **80.0$\pm$ 0.0** | **1.00$\pm$ 0.00** | -1.10$\pm$0.16 | **2.8** |
| | | $X$-means | 171.5$\pm$10.9 | 0.84$\pm$ 0.01 | **11.78$\pm$2.74** | 53.3 |

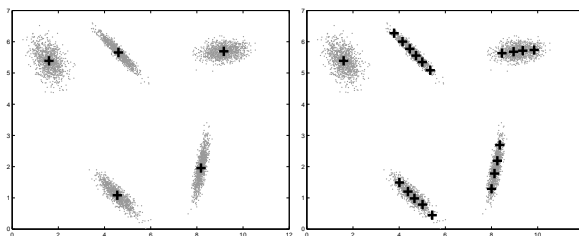

Figure 4: 2-$d$ synthetic dataset with 5 true clusters. On the left, G-means correctly chooses 5 centers and deals well with non-spherical data. On the right, the BIC causes $X$-means to overfit the data, choosing 20 unevenly distributed clusters.

well at finding the correct $k$ and maximizing the BIC statistic, so we don't show these results here. Most real-world data is not spherical, however.

The synthetic datasets used here each have 5000 datapoints in $d = 2/8/32$ dimensions. The true $k$s are 5, 20, and 80. For each synthetic dataset type, we generate 30 datasets with the true center means chosen uniformly randomly from the unit hypercube, and choosing $\sigma$ so that no two clusters are closer than $3\sigma$ apart. Each cluster is also given a transformation to make it non-spherical, by multiplying the data by a randomly chosen scaling and rotation matrix. We run G-means starting with one center. We allow $X$-means to search between 2 and $4k$ centers (where here $k$ is the true number of clusters).

The G-means algorithm clearly does better at finding the correct $k$ on non-spherical data. Its results are closer to the true distortions and the correct $k$s. The BIC statistic that $X$-means uses has been formulated to maximize the likelihood for spherically-distributed data. Thus it overestimates the number of true clusters in non-spherical data. This is especially evident when the number of points per cluster is small, as in datasets with 80 true clusters.

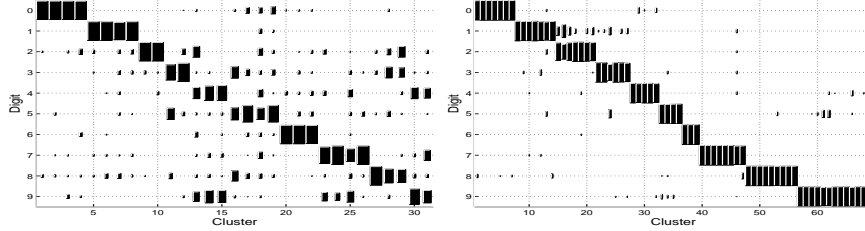

Figure 5: NIST and Pendigits datasets: correspondence between each digit (row) and each cluster (column) found by G-means. G-means did not have the labels, yet it found meaningful clusters corresponding with the labels.

Because of this overestimation, $X$-means often hits our limit of $4k$ centers. Figure 4 shows an example of overfitting on a dataset with 5 true clusters. $X$-means chooses $k = 20$ while G-means finds all 5 true cluster centers. Also of note is that $X$-means does not distribute centers evenly among clusters; some clusters receive one center, but others receive many.

G-means runs faster than $X$-means for 8 and 32 dimensions, which we expect, since the $kd$-tree structures which make $X$-means fast in low dimensions take time exponential in $d$, making them slow for more than 8 to 12 dimensions. All our code is written in Matlab; $X$-means is written in C.

### 3.1   Discovering true clusters in labeled data

We tested these algorithms on two real-world datasets for handwritten digit recognition: the NIST dataset [12] and the Pendigits dataset [2]. The goal is to cluster the data without knowledge of the labels and measure how well the clustering captures the true labels. Both datasets have 10 true classes (digits 0-9). NIST has 60000 training examples and 784 dimensions (28×28 pixels). We use 6000 randomly chosen examples and we reduce the dimension to 50 by random projection (following [3]). The Pendigits dataset has 7984 examples and 16 dimensions; we did not change the data in any way.

We cluster each dataset with G-means and $X$-means, and measure performance by comparing the cluster labels $L_c$ with the true labels $L_t$. We define the *partition quality* (PQ) as $pq = \sum_{i=1}^{k_t} \sum_{j=1}^{k_c} p(i,j)^2 \Big/ \sum_{i=1}^{k_t} p(i)^2$ where $k_t$ is the true number of classes, and $k_c$ is the number of clusters found by the algorithm. This metric is maximized when $L_c$ induces the same partition of the data as $L_t$; in other words, when all points in each cluster have the same true label, and the estimated $k$ is the true $k$. The $p(i,j)$ term is the frequency-based probability that a datapoint will be labeled $i$ by $L_t$ and $j$ by $L_c$. This quality is normalized by the sum of true probabilities, squared. This statistic is related to the Rand statistic for comparing partitions [8].

For the NIST dataset, G-means finds 31 clusters in 30 seconds with a PQ score of 0.177. $X$-means finds 715 clusters in 4149 seconds, and 369 of these clusters contain only one point, indicating an overestimation problem with the BIC. $X$-means receives a PQ score of 0.024. For the Pendigits dataset, G-means finds 69 clusters in 30 seconds, with a PQ score of 0.196; $X$-means finds 235 clusters in 287 seconds, with a PQ score of 0.057. Figure 5 shows Hinton diagrams of the G-means clusterings of both datasets, showing that G-means succeeds at identifying the true clusters concisely, without aid of the labels. The confusions between different digits in the NIST dataset (seen in the off-diagonal elements) are common for other researchers using more sophisticated techniques, see [3].

# 4  Discussion and conclusions

We have introduced the new G-means algorithm for learning $k$ based on a statistical test for determining whether datapoints are a random sample from a Gaussian distribution with arbitrary dimension and covariance matrix. The splitting uses dimension reduction and a powerful test for Gaussian fitness. G-means uses this statistical test as a wrapper around $k$-means to discover the number of clusters automatically. The only parameter supplied to the algorithm is the significance level of the statistical test, which can easily be set in a standard way. The G-means algorithm takes linear time and space (plus the cost of the splitting heuristic and test) in the number of datapoints and dimension, since $k$-means is itself linear in time and space. Empirically, the G-means algorithm works well at finding the correct number of clusters and the locations of genuine cluster centers, and we have shown it works well in moderately high dimensions.

Clustering in high dimensions has been an open problem for many years. Recent research has shown that it may be preferable to use dimensionality reduction techniques before clustering, and then use a low-dimensional clustering algorithm such as $k$-means, rather than clustering in the high dimension directly. In [3] the author shows that using a simple, inexpensive linear projection preserves many of the properties of data (such as cluster distances), while making it easier to find the clusters. Thus there is a need for good-quality, fast clustering algorithms for low-dimensional data. Our work is a step in this direction.

Additionally, recent image segmentation algorithms such as normalized cut [16, 13] are based on eigenvector computations on distance matrices. These "spectral" clustering algorithms still use $k$-means as a post-processing step to find the actual segmentation and they require $k$ to be specified. Thus we expect G-means will be useful in combination with spectral clustering.

## References

[1] Horst Bischof, Aleš Leonardis, and Alexander Selb. MDL principle for robust vector quantisation. *Pattern analysis and applications*, 2:59–72, 1999.

[2] C.L. Blake and C.J. Merz. UCI repository of machine learning databases, 1998. http://www.ics.uci.edu/~mlearn/MLRepository.html.

[3] Sanjoy Dasgupta. Experiments with random projection. In *Uncertainty in Artificial Intelligence: Proceedings of the Sixteenth Conference (UAI-2000)*, pages 143–151, San Francisco, CA, 2000. Morgan Kaufmann Publishers.

[4] Gianna M. Del Corso. Estimating an eigenvector by the power method with a random start. *SIAM Journal on Matrix Analysis and Applications*, 18(4):913–937, 1997.

[5] Chris Ding, Xiaofeng He, Hongyuan Zha, and Horst Simon. Adaptive dimension reduction for clustering high dimensional data. In *Proceedings of the 2nd IEEE International Conference on Data Mining*, 2002.

[6] Fredrik Farnstrom, James Lewis, and Charles Elkan. Scalability for clustering algorithms revisited. *SIGKDD Explorations*, 2(1):51–57, 2000.

[7] Peter J. Huber. Projection pursuit. *Annals of Statistics*, 13(2):435–475, June 1985.

[8] L. Hubert and P. Arabie. Comparing partitions. *Journal of Classification*, 2:193–218, 1985.

[9] A. K. Jain, M. N. Murty, and P. J. Flynn. Data clustering: a review. *ACM Computing Surveys*, 31(3):264–323, 1999.

[10] Robert E. Kass and Larry Wasserman. A reference Bayesian test for nested hypotheses and its relationship to the Schwarz criterion. *Journal of the American Statistical Association*, 90(431):928–934, 1995.

[11] Michael J. Kearns, Yishay Mansour, Andrew Y. Ng, and Dana Ron. An experimental and theoretical comparison of model selection methods. In *Computational Learing Theory (COLT)*, pages 21–30, 1995.

[12] Yann LeCun, Léon Bottou, Yoshua Bengio, and Patrick Haffner. Gradient-based learning applied to document recognition. *Proceedings of the IEEE*, 86(11):2278–2324, 1998.

[13] Andrew Ng, Michael Jordan, and Yair Weiss. On spectral clustering: Analysis and an algorithm. *Neural Information Processing Systems*, 14, 2002.

[14] Dan Pelleg and Andrew Moore. $X$-means: Extending $K$-means with efficient estimation of the number of clusters. In *Proceedings of the 17th International Conf. on Machine Learning*, pages 727–734. Morgan Kaufmann, San Francisco, CA, 2000.

[15] Peter Sand and Andrew Moore. Repairing faulty mixture models using density estimation. In *Proceedings of the 18th International Conf. on Machine Learning*. Morgan Kaufmann, San Francisco, CA, 2001.

[16] Jianbo Shi and Jitendra Malik. Normalized cuts and image segmentation. *IEEE Transactions on Pattern Analysis and Machine Intelligence*, 22(8):888–905, 2000.

[17] M. A. Stephens. EDF statistics for goodness of fit and some comparisons. *American Statistical Association*, 69(347):730–737, September 1974.
